# Softassign versus Softmax: Benchmarks in Combinatorial Optimization

**Steven Gold**
Department of Computer Science
Yale University
New Haven, CT 06520-8285

**Anand Rangarajan**
Dept. of Diagnostic Radiology
Yale University
New Haven, CT 06520-8042

## Abstract

A new technique, termed *softassign*, is applied for the first time to two classic combinatorial optimization problems, the traveling salesman problem and graph partitioning. Softassign, which has emerged from the recurrent neural network/statistical physics framework, enforces two-way (assignment) constraints without the use of penalty terms in the energy functions. The softassign can also be generalized from two-way winner-take-all constraints to multiple membership constraints which are required for graph partitioning. The softassign technique is compared to the softmax (Potts glass). Within the statistical physics framework, softmax and a penalty term has been a widely used method for enforcing the two-way constraints common within many combinatorial optimization problems. The benchmarks present evidence that softassign has clear advantages in accuracy, speed, parallelizability and algorithmic simplicity over softmax and a penalty term in optimization problems with two-way constraints.

## 1 Introduction

In a series of papers in the early to mid 1980's, Hopfield and Tank introduced techniques which allowed one to solve combinatorial optimization problems with recurrent neural networks [Hopfield and Tank, 1985]. As researchers attempted to reproduce the original traveling salesman problem results of Hopfield and Tank, problems emerged, especially in terms of the quality of the solutions obtained. More recently however, a number of techniques from statistical physics have been adopted to mitigate these problems. These include deterministic annealing which convexifies the energy function in order help avoid some local minima and the Potts glass approximation which results in a hard enforcement of a one-way (one set of) winner-take-all (WTA) constraint via the softmax. In

the late 80's, armed with these techniques optimization problems like the traveling salesman problem (TSP) [Peterson and Soderberg, 1989] and graph partitioning [Peterson and Soderberg, 1989, Van den Bout and Miller III, 1990] were reexamined and much better results compared to the original Hopfield-Tank dynamics were obtained.

However, when the problem calls for two-way interlocking WTA constraints, as do TSP and graph partitioning, the resulting energy function must still include a penalty term when the softmax is employed in order to enforce the second set of WTA constraints. Such penalty terms may introduce spurious local minima in the energy function and involve free parameters which are hard to set. A new technique, termed *softassign*, eliminates the need for all such penalty terms. The first use of the softassign was in an algorithm for the assignment problem [Kosowsky and Yuille, 1994]. It has since been applied to much more difficult optimization problems, including parametric assignment problems—point matching [Gold et al., 1994, Gold et al., 1995, Gold et al., 1996] and quadratic assignment problems—graph matching [Gold et al., 1996, Gold and Rangarajan, 1996, Gold, 1995].

Here, we for the first time apply the softassign to two classic combinatorial optimization problems, TSP and graph partitioning. Moreover, we show that the softassign can be generalized from two-way winner-take-all constraints to multiple membership constraints, which are required for graph partitioning (as described below). We then run benchmarks against the older softmax (Potts glass) methods and demonstrate advantages in terms of accuracy, speed, parallelizability, and simplicity of implementation.

It must be emphasized there are other conventional techniques, for solving some combinatorial optimization problems such as TSP, which remain superior to this method in certain ways [Lawler et al., 1985]. (We think for some problems—specifically the type of pattern matching problems essential for cognition [Gold, 1995]—this technique is superior to conventional methods.) Even within neural networks, elastic net methods may still be better in certain cases. However, the elastic net uses only a one-way constraint in TSP. The main goal of this paper is to provide evidence, that when minimizing energy functions within the neural network framework, which have two-way constraints, the softassign should be the technique of choice. We therefore compare it to the current dominant technique, softmax with a penalty term.

## 2   Optimizing With Softassign

### 2.1   The Traveling Salesman Problem

The traveling salesman problem may be defined in the following way. Given a set of intercity distances $\{\delta_{ab}\}$ which may take values in $R^+$, find the permutation matrix $M$ such that the following objective function is minimized.

$$E_1(M) = \frac{1}{2} \sum_{a=1}^{N} \sum_{b=1}^{N} \sum_{i=1}^{N} \delta_{ab} M_{ai} M_{b(i \oplus 1)} \tag{1}$$

subject to $\forall a \ \sum_{i=1}^{N} M_{ai} = 1$ , $\forall i \ \sum_{a=1}^{N} M_{ai} = 1$ , $\forall ai \ M_{ai} \in \{0, 1\}$.

In the above objective $\delta_{ab}$ represents the distance between cities $a$ and $b$. $M$ is a permutation matrix whose rows represent cities, and whose columns represent the day (or order) the city was visited and $N$ is the number of cities. (The notation $i \oplus 1$

is used to indicate that subscripts are defined modulo $N$, i.e. $M_{a(N+1)} = M_{a1}$.) So if $M_{ai} = 1$ it indicates that city $a$ was visited on day $i$.

Then, following [Peterson and Soderberg, 1989, Yuille and Kosowsky, 1994] we employ Lagrange multipliers and an $x \log x$ barrier function to enforce the constraints, as well as a $\gamma$ term for stability, resulting in the following objective:

$$E_2(M, \mu, \nu) = \frac{1}{2} \sum_{a=1}^{N} \sum_{b=1}^{N} \sum_{i=1}^{N} \delta_{ab} M_{ai} M_{b(i \oplus 1)} - \frac{\gamma}{2} \sum_{a=1}^{N} \sum_{i=1}^{N} M_{ai}^2$$

$$+ \frac{1}{\beta} \sum_{a=1}^{N} \sum_{i=1}^{N} M_{ai}(\log M_{ai} - 1) + \sum_{a=1}^{N} \mu_a (\sum_{i=1}^{N} M_{ai} - 1) + \sum_{i=1}^{N} \nu_i (\sum_{a=1}^{N} M_{ai} - 1) \qquad (2)$$

In the above we are looking for a saddle point by minimizing with respect to $M$ and maximizing with respect to $\mu$ and $\nu$, the Lagrange multipliers.

## 2.2   The Softassign

In the above formulation of TSP we have two-way interlocking WTA constraints. $\{M_{ai}\}$ must be a permutation matrix to ensure that a valid tour—one in which each city is visited once and only once—is described. A permutation matrix means all the rows and columns must add to one (and the elements must be zero or one) and therefore requires two-way WTA constraints—a set of WTA constraints on the rows and a set of WTA constraints on the columns. This set of two-way constraints may also be considered assignment constraints, since each city must be assigned to one and only one day (the row constraint) and each day must be assigned to one and only one city (the column constraint).

These assignment constraints can be satisfied using a result from [Sinkhorn, 1964]. In [Sinkhorn, 1964] it is proven that any square matrix whose elements are all positive will converge to a doubly stochastic matrix just by the iterative process of alternatively normalizing the rows and columns. (A doubly stochastic matrix is a matrix whose elements are all positive and whose rows and columns all add up to one—it may roughly be thought of as the continuous analog of a permutation matrix).

The softassign simply employs Sinkhorn's technique within a deterministic annealing context. Figure 1 depicts the contrast between the softassign and the softmax. In the softmax, a one-way WTA constraint is strictly enforced by normalizing over a vector.

[Kosowsky and Yuille, 1994] used the softassign to solve the assignment problem, i.e. minimize: $-\sum_{a=1}^{A} \sum_{i=1}^{I} M_{ai} Q_{ai}$. For the special case of the quadratic assignment problem, being solved here, by setting $Q_{ai} = -\frac{\partial \dot{E}_2}{\partial M_{ai}}$, and using the values of $M$ from the previous iteration, we can at each iteration produce a new assignment problem for which the softassign then returns a doubly stochastic matrix. As the temperature is lowered a series of assignment problems are generated, along with the corresponding doubly stochastic matrices returned by each softassign, until a permutation matrix is reached.

The update with the partial derivative in the preceding may be derived using a Taylor series expansion. See [Gold and Rangarajan, 1996, Gold, 1995] for details.

The algorithm dynamics then become:

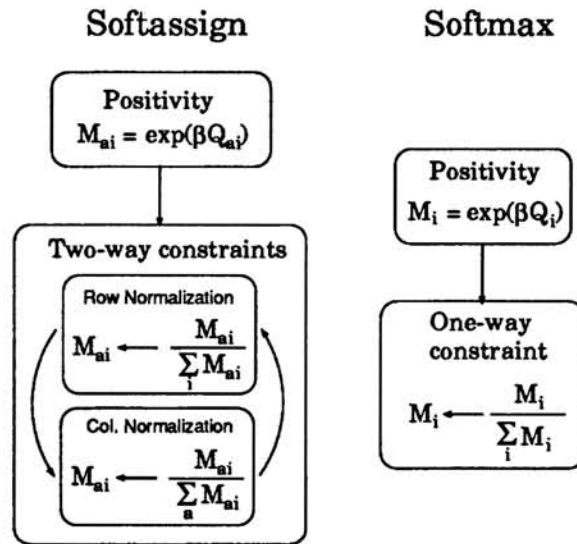

Figure 1: Softassign and softmax. This paper compares these two techniques.

$$Q_{ai} = -\frac{\partial \hat{E}_2}{\partial M_{ai}} \tag{3}$$

$$M_{ai} = Softassign_{ai}(Q) \tag{4}$$

$\hat{E}_2$ is $E_2$ without the $\beta$, $\mu$ or $\nu$ terms of (2), therefore no penalty terms are now included. The above dynamics are iterated as $\beta$, the inverse temperature, is gradually increased.

These dynamics may be obtained by evaluating the saddle points of the objective in (2). Sinkhorn's method finds the saddle points for the Lagrange parameters.

## 2.3   Graph Partitioning

The graph partitioning problem maybe defined in the following way. Given an unweighted graph $G$, find the membership matrix $M$ such that the following objective function is minimized.

$$E_3(M) = -\sum_{a=1}^{A}\sum_{i=1}^{I}\sum_{j=1}^{I} G_{ij} M_{ai} M_{aj} \tag{5}$$

subject to $\forall a \sum_{i=1}^{I} M_{ai} = I/A$, $\forall i \sum_{a=1}^{A} M_{ai} = 1$, $\forall ai\ M_{ai} \in \{0,1\}$ where graph $G$ has $I$ nodes which should be equally partitioned into $A$ bins.

$\{G_{ij}\}$ is the adjacency matrix of the graph, whose elements must be 0 or 1. $M$ is a membership matrix such that $M_{ai} = 1$ indicates that node $i$ is in bin $a$. The permutation matrix constraint present in TSP is modified to the membership constraint. Node $i$ is a member of only bin $a$ and the number of members in each bin is fixed at $I/A$. When the above objective is at a minimum, then graph $G$ will be partitioned into $A$ equal sized bins, such that the cutsize is minimum for all possible partitionings of $G$ into $A$ equal sized bins. We assume $I/A$ is an integer.

Then following the treatment for TSP, we derive the following objective:

$$E_4(M, \mu, \nu) = - \sum_{a=1}^{A} \sum_{i=1}^{I} \sum_{j=1}^{I} G_{ij} M_{ai} M_{aj} - \frac{\gamma}{2} \sum_{a=1}^{A} \sum_{i=1}^{I} M_{ai}^2$$

$$+ \frac{1}{\beta} \sum_{a=1}^{A} \sum_{i=1}^{I} M_{ai} (\log M_{ai} - 1) + \sum_{a=1}^{A} \mu_a (\sum_{i=1}^{I} M_{ai} - I/A) + \sum_{i=1}^{I} \nu_i (\sum_{a=1}^{A} M_{ai} - 1) \quad (6)$$

which is minimized with a similar algorithm employing the softassign. Note however now in the softassign the columns are normalized to $I/A$ instead of 1.

## 3   Experimental Results

Experiments on Euclidean TSP and graph partitioning were conducted. For each problem three different algorithms were run. One used the softassign described above. The second used the Potts glass dynamics employing synchronous update as described in [Peterson and Soderberg, 1989]. The third used the Potts glass dynamics employing serial update as described in [Peterson and Soderberg, 1989]. Originally the intention was to employ just the synchronous updating version of the Potts glass dynamics, since that is the dynamics used in the algorithms employing softassign and is the method that is massively parallelizable. We believe massive parallelism to be such a critical feature of the neural network architecture [Rumelhart and McClelland, 1986] that any algorithm that does not have this feature loses much of the power of the neural network paradigm. Unfortunately the synchronous updating algorithms just worked so poorly that we also ran the serial versions in order to get a more extensive comparison. Note that the results reported in [Peterson and Soderberg, 1989] were all with the serial versions.

### 3.1   Euclidean TSP Experiments

Figure 2 shows the results of the Euclidean TSP experiments. 500 different 100-city tours from points uniformly generated in the 2D unit square were used as input. The asymptotic expected length of an optimal tour for cities distributed in the unit square is given by $L(n) = K\sqrt{n}$ where $n$ is the number of cities and $0.765 \leq K \leq 0.765 + \frac{4}{n}$ [Lawler et al., 1985]. This gives the interval $[7.65, 8.05]$ for the 100 city TSP. 95% of the tour lengths fall in the interval $[8, 11]$ when using the softassign approach. Note the large difference in performance between the softassign and the Potts glass algorithms. The serial Potts glass algorithm ran about 5 times slower than the softassign version. Also as noted previously the serial version is not massively parallelizable. The synchronous Potts glass ran about 2 times slower. Also note the softassign algorithm is much simpler to implement—fewer parameters to tune.

### 3.2   Graph Partitioning Experiments

Figure 3 shows the results of the graph partitioning experiments. 2000 different randomly generated 100 node graphs with 10% connectivity were used as input. These graphs were partitioned into four bins. The softassign performs better than the Potts glass algorithms, however here the difference is more modest than in the TSP experiments. However the serial Potts glass algorithm again ran about 5 times slower then the softassign version and as noted previously the serial version is not massively parallelizable. The synchronous Potts glass ran about 2 times slower.

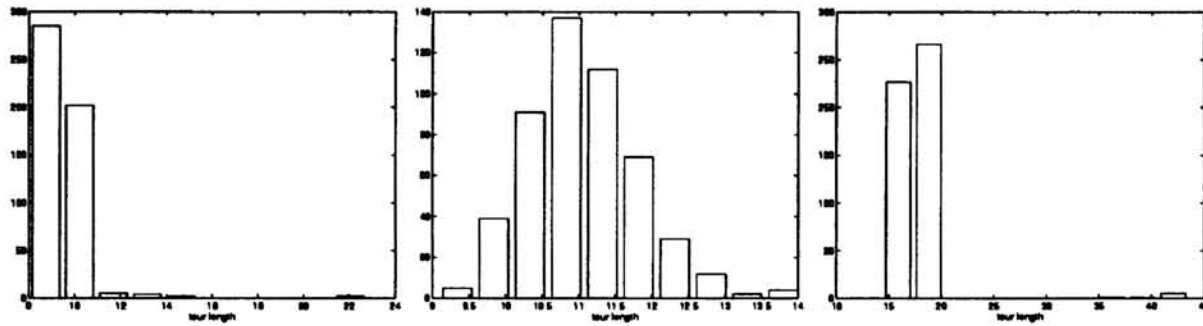

Figure 2: 100 City Euclidean TSP. 500 experiments. Left: **Softassign.**. Middle: **Softmax (serial update)**. Right: **Softmax (synchronous update)**.

Also again note the softassign algorithm was much simpler to implement—fewer parameters to tune.

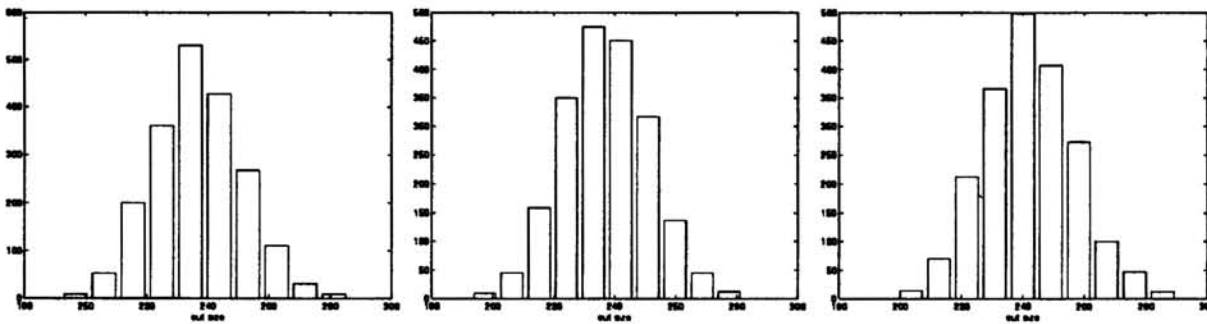

Figure 3: 100 node Graph Partitioning, 4 bins. 2000 experiments. Left: **Softassign.**. Middle: **Softmax (serial update)**. Right: **Softmax (synchronous update)**.

A relatively simple version of graph partitioning was run. It is likely that as the number of bins are increased the results on graph partitioning will come to resemble more closely the TSP results, since when the number of bins equal the number of nodes, the TSP can be considered a special case of graph partitioning (there are some additional restrictions). However even in this simple case the softassign has clear advantages over the softmax and penalty term.

## 4    Conclusion

For the first time, two classic combinatorial optimization problems, TSP and graph partitioning, are solved using a new technique for constraint satisfaction, the softassign. The softassign, which has recently emerged from the statistical physics/neural networks framework, enforces a two-way (assignment) constraint, without penalty terms in the energy function. We also show that the softassign can be generalized from two-way winner-take-all constraints to multiple membership constraints, which are required for graph partitioning. Benchmarks against the Potts glass methods, using softmax and a penalty term, clearly demonstrate its advantages in terms of accuracy, speed, parallelizability and simplicity of implementation. Within the neural network/statistical physics framework, softassign should be considered the technique of choice for enforcing two-way constraints in energy functions.

# References

[Gold, 1995] Gold, S. (1995). *Matching and Learning Structural and Spatial Representations with Neural Networks*. PhD thesis, Yale University.

[Gold et al., 1995] Gold, S., Lu, C. P., Rangarajan, A., Pappu, S., and Mjolsness, E. (1995). New algorithms for 2-D and 3-D point matching: pose estimation and correspondence. In Tesauro, G., Touretzky, D. S., and Leen, T. K., editors, *Advances in Neural Information Processing Systems 7*, pages 957–964. MIT Press, Cambridge, MA.

[Gold et al., 1994] Gold, S., Mjolsness, E., and Rangarajan, A. (1994). Clustering with a domain specific distance measure. In Cowan, J., Tesauro, G., and Alspector, J., editors, *Advances in Neural Information Processing Systems 6*, pages 96–103. Morgan Kaufmann, San Francisco, CA.

[Gold and Rangarajan, 1996] Gold, S. and Rangarajan, A. (1996). A graduated assignment algorithm for graph matching. *IEEE Transactions on Pattern Analysis and Machine Intelligence*, (in press).

[Gold et al., 1996] Gold, S., Rangarajan, A., and Mjolsness, E. (1996). Learning with preknowledge: clustering with point and graph matching distance measures. *Neural Computation*, (in press).

[Hopfield and Tank, 1985] Hopfield, J. J. and Tank, D. (1985). 'Neural' computation of decisions in optimization problems. *Biological Cybernetics*, 52:141–152.

[Kosowsky and Yuille, 1994] Kosowsky, J. J. and Yuille, A. L. (1994). The invisible hand algorithm: Solving the assignment problem with statistical physics. *Neural Networks*, 7(3):477–490.

[Lawler et al., 1985] Lawler, E. L., Lenstra, J. K., Kan, A. H. G. R., and Shmoys, D. B., editors (1985). *The Traveling Salesman Problem*. John Wiley and Sons, Chichester.

[Peterson and Soderberg, 1989] Peterson, C. and Soderberg, B. (1989). A new method for mapping optimization problems onto neural networks. *Intl. Journal of Neural Systems*, 1(1):3–22.

[Rumelhart and McClelland, 1986] Rumelhart, D. and McClelland, J. L. (1986). *Parallel Distributed Processing*, volume 1. MIT Press, Cambridge, MA.

[Sinkhorn, 1964] Sinkhorn, R. (1964). A relationship between arbitrary positive matrices and doubly stochastic matrices. *Ann. Math. Statist.*, 35:876–879.

[Van den Bout and Miller III, 1990] Van den Bout, D. E. and Miller III, T. K. (1990). Graph partitioning using annealed networks. *IEEE Trans. Neural Networks*, 1(2):192–203.

[Yuille and Kosowsky, 1994] Yuille, A. L. and Kosowsky, J. J. (1994). Statistical physics algorithms that converge. *Neural Computation*, 6(3):341–356.